# Differentiable Sparse Coding

**David M. Bradley**
Robotics Institute
Carnegie Mellon University
Pittsburgh, PA 15213
dbradley@cs.cmu.edu

**J. Andrew Bagnell**
Robotics Institute
Carnegie Mellon University
Pittsburgh, PA 15213
dbagnell@ri.cmu.edu

## Abstract

Prior work has shown that features which appear to be biologically plausible as well as empirically useful can be found by sparse coding with a prior such as a laplacian ($L_1$) that promotes sparsity. We show how smoother priors can preserve the benefits of these sparse priors while adding stability to the Maximum A-Posteriori (MAP) estimate that makes it more useful for prediction problems. Additionally, we show how to calculate the derivative of the MAP estimate efficiently with implicit differentiation. One prior that can be differentiated this way is KL-regularization. We demonstrate its effectiveness on a wide variety of applications, and find that online optimization of the parameters of the KL-regularized model can significantly improve prediction performance.

## 1 Introduction

Sparse approximation is a key technique developed in engineering and the sciences which approximates an input signal, $X$, in terms of a "sparse" combination of fixed bases $B$. Sparse approximation relies on an *optimization* algorithm to infer the Maximum A-Posteriori (MAP) weights $\hat{W}$ that best reconstruct the signal, given the model $X \approx f(BW)$. In this notation, each input signal forms a column of an input matrix $X$, and is generated by multiplying a set of basis vectors $B$, and a column from a coefficient matrix $W$, while $f(z)$ is an optional transfer function. This relationship is only approximate, as the input data is assumed to be corrupted by random noise. Priors which produce sparse solutions for $W$, especially $L_1$ regularization, have gained attention because of their usefulness in ill-posed engineering problems [1], their ability to elucidate certain neuro-biological phenomena, [2, 3], and their ability to identify useful features for classification from related unlabeled data [4].

Sparse coding [2] is closely connected to Independent Component Analysis as well as to certain approaches to matrix factorization. It extends sparse approximation by learning a basis matrix $B$ which represents well a collection of related input signals–the input matrix $X$–in addition to performing optimization to compute the best set of weights $\hat{W}$. Unfortunately, existing sparse coding algorithms that leverage an efficient, convex sparse approximation step to perform inference on the latent weight vector [4] are difficult to integrate into a larger learning architecture. It has been convincingly demonstrated that back-propagation is a crucial tool for tuning an existing generative model's output in order to improve supervised performance on a discriminative task. For example, greedy layer-wise strategies for building deep generative models rely upon a back-propagation step to achieve excellent model performance [5]. Unfortunately, existing sparse coding architectures produce a latent representation $\hat{W}$ that is an unstable, *discontinuous* function of the inputs and bases; an arbitrarily small change in input can lead to the selection of a completely different set of latent weights.

We present an advantageous new approach to coding that uses smoother priors which preserve the sparsity benefits of $L_1$-regularization while allowing efficient convex inference and producing stable latent representations $\hat{W}$. In particular we examine a prior based on minimizing KL-divergence to

the uniform distribution which has long been used for approximation problems [6, 7]. We show this increased stability leads to better semi-supervised classification performance across a wide variety of applications for classifiers using the latent representation $\hat{W}$ as input. Additionally, because of the smoothness of the KL-divergence prior, $B$ can be optimized discriminatively for a particular application by gradient descent, leading to outstanding empirical performance.

## 2  Notation

Uppercase letters, $X$, denote matrices and lowercase letters, $x$, denote vectors. For matrices, superscripts and subscripts denote rows and columns respectively. $X_j$ is the jth column of $X$, $X^i$ is the ith row of $X$, and $X_j^i$ is the element in the ith row and jth column. Elements of vectors are indicated by subscripts, $x_j$, and superscripts on vectors are used for time indexing $x^t$. $X^T$ is the transpose of matrix $X$.

## 3  Generative Model

Sparse coding fits a generative model (1) to unlabeled data, and the MAP estimates of the latent variables of this model have been shown to be useful as input for prediction problems [4]. (1) divides the latent variables into two independent groups, the coefficients $W$ and the basis $B$, which combine to form the matrix of input examples $X$. Different examples (columns of $X$) are assumed to be independent of each other. The Maximum A Posteriori (MAP) approximation replaces the integration over $W$ and $B$ in (1) with the maximum value of $P(X|W, B)P(W)P(B)$, and the values of the latent variables at the maximum, $\hat{W}$ and $\hat{B}$, are the MAP estimates.

Finding $\hat{W}$ given $B$ is an approximation problem, solving for $\hat{W}$ and $\hat{B}$ simultaneously over a set of independent examples is a coding problem.

$$P(X) = \int_B \int_W P(X|W,B)P(W)P(B)dW dB = \int_B P(B)\int_W \prod_i P(X_i|W_i,B)P(W_i)dW dB \quad (1)$$

Given $B$, the negative log of the generative model can be optimized independently for each example, and it is denoted for a generic example $x$ by $\mathfrak{L}$ in (2). $\mathfrak{L}$ decomposes into the sum of two terms, a loss function $D_L\left(x\|f(Bw)\right)$ between an input example and the reconstruction produced by the transfer function $f$, and a regularization function $D_P(w\|p)$ that measures a distance between the coefficients for the example $w$ and a parameter vector $p$. A regularization constant $\lambda$ controls the relative weight of these two terms. For fixed $B$, minimizing (2) with respect to $w$ separately for each example is equivalent to maximizing (1).

$$\mathfrak{L} = D_L\left(x\|f(Bw)\right) + \lambda D_P(w\|p) \quad (2)$$
$$\hat{w} = \arg\min_w \mathfrak{L} \quad (3)$$

In many applications, the anticipated distribution of $x$ after being corrupted by noise can be modeled by an exponential family distribution. Every exponential family distribution defines a Bregman divergence which serves as a matching loss function for estimating the parameters of the distribution[1]. One common choice for the loss/transfer functions is the squared loss function with its matching linear transfer function, $D_L\left(x\|f(Bw)\right) = \sum_i(x_i - B^iw)^2$, which is the matching Bregman Divergence for $x$ drawn from a multidimensional gaussian distribution.

The regularization function $D_P(w\|p)$ is also often a Bregman divergence, but may be chosen for other features such as the sparsity of the resulting MAP estimate $\hat{w}$. A vector is commonly called sparse if many elements are exactly zero. The entropy [9, 10], and $L_p^p$-norm[2], $p \le 1$ regularization functions [2, 3, 4] promote this form of sparsity, and all of them have shown the ability to learn bases

containing interesting structure from unlabeled data. However, of these only $L_1$ leads to an efficient, convex procedure for inference, and even this prior does not produce differentiable MAP estimates.

We argue that if the latent weight vector $\hat{w}$ is to be used as input to a classifier, a better definition of "sparsity" is that most elements in $\hat{w}$ can be replaced by elements in a constant vector $p$ without significantly increasing the loss. One regularization function that produces this form of pseudo-sparsity is the KL-divergence $KL(w\|p)$. This regularization function has long been used for approximation problems in Geophysics, Crystallography, Astronomy, and Physics, where it is commonly referred to as Maximum Entropy on the Mean (MEM) [7], and has been shown in the online setting to compete with low $L_1$-norm solutions in terms of regret [11, 12].

$L_1$ regularization provides sparse solutions because its Fenchel dual [13] is the max function, meaning only the most useful basis vectors participate in the reconstruction. A differentiable approximation to $\max_i x_i$ is a sum of exponentials, $\sum_i e_i^x$, whose dual is the KL-divergence (4). Regularization with KL has proven useful in online learning, where it is the implicit prior of the exponentiated gradient descent (EGD) algorithm. EGD has been shown to be "sparse" in the sense that it can select a few relevant features to use for a prediction task from many irrelevant ones.

The form of KL we use (4) is the full Bregman divergence of the negative entropy function[3]. Often KL is used to compute distances between probability distributions, and for this case the KL we use reduces to the standard form. For sparse coding however, it is inconvenient to assume that $\|\hat{w}\|_1 = \|p\|_1 = 1$, so we use the full unnormalized KL instead.

$$D_P(w\|p) = \sum_i \left[ w_i \log \frac{w_i}{p_i} - w_i + p_i \right] \qquad (4)$$

For the prior vector $p$ we use a uniform vector whose $L_1$ magnitude equals the expected $L_1$ magnitude of $w$. $p$ has an analogous effect to the $q$ parameter in $L_q$-norm regularization. $p \to 0$ approximates $L_1$ and $p \to \infty$ approximates $L_2$. Changing $p$ affects the magnitude of the KL term, so $\lambda$ in (2) must be adjusted to balance the loss term in the sparse coding objective function (small values of $p$ require small values of $\lambda$).

Below we provide a) an efficient procedure for inferring $\hat{w}$ in this model; b) an algorithm for iteratively updating the bases $B$, and c) show that this model leads to differentiable estimates of $\hat{w}$. We also provide the general form of the derivative for arbitrary Bregman losses.

## 4  Implementation

To compute $\hat{w}$ with KL-regularization, we minimize (3) using exponentiated gradient descent (EGD) with backtracking until convergence (5). EGD automatically enforces positivity constraints on the coefficient vector $w$, and is particularly efficient for optimization because it is the natural mirror descent rule for KL-regularization [12]. The gradient of the objective function (2) with respect to the coefficient for the jth basis vector $w_j$ is given in (6) for matching loss/transfer function pairs.

$$w_j^{t+1} = w_j^t e^{-\alpha \frac{\partial \mathfrak{L}}{\partial w_j}} \qquad (5)$$

$$\frac{\partial \mathfrak{L}}{\partial w_j} = (f(Bw) - x)^T B_j + \lambda \log \frac{w_j}{p_j} \qquad (6)$$

This iterative update is run until the maximum gradient element is less than a threshold, which is estimated by periodically running a random set of examples to the limits of machine precision, and selecting the largest gradient threshold that produces $\hat{w}$ within $\epsilon$ of the exact solution. The $\alpha$ parameter is continuously updated to balance the number of sucessful steps and the number of backtracking steps[4]. Because $L_1$-regularization produces both positive and negative weights, to compare $L_1$ and KL regularization on the same basis we expand the basis used for KL by adding the negation of each basis vector, which is equivalent to allowing negative weights (see Appendix B).

During sparse coding the basis matrix $B$ is updated by Stochastic Gradient Descent (SGD), giving the update rule $B_{t+1} = B_t - \eta \frac{\partial L}{\partial B_j^i}$. This update equation does not depend on the prior chosen

for $w$ and is given in (7) for matching loss/transfer function pairs. SGD implements an implicit $L_2$ regularizer and is suitable for online learning, however because the magnitude of $w$ is explicitly penalized, the columns of $B$ were constrained to have unit $L_2$ norm to prevent the trivial solution of infinitely large $B$ and infinitely small $w$. The step size was adjusted for the magnitude of $\hat{w}$ in each application, and was then decayed over time as $\eta \propto 1/\sqrt{t}$. The same SGD procedure was also used to optimize $B$ through backpropagation, as explained in the next section.

$$\frac{\partial L}{\partial B_j^i} = w_j(f(B^i w) - x_i) \tag{7}$$

## 5  Modifying a Generative Model For A Discriminative Task

Sparse Coding builds a generative model from unlabeled data that captures structure in that data by learning a basis $B$. Our hope is that the MAP estimate of basis coefficients $\hat{w}$ produced for each input vector $x$ will be useful for predicting a response $y$ associated with $x$. However, the sparse coding objective function only cares about reconstructing the input well, and does not attempt to make $\hat{w}$ useful as input for any particular task. Fortunately, since priors such as KL-divergence regularization produce solutions that are smooth with respect to small changes in $B$ and $x$, $B$ can be modified through back-propagation to make $\hat{w}$ more useful for prediction.

The key to computing the derivatives required for backpropagation is noting that the gradient with respect to $w$ of the optimization (3) at its minimum $\hat{w}$ can be written as a set of fixed point equations where the gradients of the loss term equal the gradient of the regularization:

$$\nabla D_P(\hat{w}\|p) = -\frac{1}{\lambda}\nabla D_L\left(x\|f(B\hat{w})\right). \tag{8}$$

Then if the regularization function is twice differentiable with respect to $w$, we can use *implicit differentiation* on (8) to compute the gradient of $\hat{w}$ with respect to $B$, and $x$ [14]. For KL-regularization and the simple case of a linear transfer function with squared loss, $\frac{\partial \hat{w}}{\partial B}$ is given in (9), where $\vec{e}_i$ is a unit vector whose ith element is 1. A general derivation for matched loss/transfer function pairs as defined before is provided in appendix C. Note that the ability to compute $\frac{\partial \hat{w}}{\partial x}$ means that multiple layers of sparse coding could be used.

$$\frac{\partial \hat{w}}{\partial B_i^k} = -\left(B^T B + \mathrm{diag}(\frac{\lambda}{\hat{w}})\right)^{-1}\left((B^k \hat{w}_i)^T + \vec{e}_i(f(B^k \hat{w}) - x_k)\right) \tag{9}$$

## 6  Experiments

We verify the performance of KL-sparse coding on several benchmark tasks including the MNIST handwritten digit recognition data-set, handwritten lowercase English characters classification, movie review sentiment regression, and music genre classification (Appendix E). In each application, the $\hat{w}$ produced using KL-regularization were more useful for prediction than those produced with $L_1$ regularization due to the stability and differentiability provided by KL.

### 6.1  Sparsity

KL-regularization retained the desirable pseudo-sparsity characteristics of $L_1$, namely that each example, $x$, produces only a few large elements in $\hat{w}$. Figure 1 compares the mean sorted and normalized coefficient distribution over the 10,000 digit MNIST test set for KL-divergence and several $L_p^p$ regularization functions, and shows that although the KL regularization function is not sparse in the traditional sense of setting many elements of $\hat{w}$ to zero, it is sparse in the sense that $\hat{w}$ contains only a few large elements in each example, lending support to the idea that this sense of sparsity is more important for classification.

### 6.2  Stability

Because the gradient of the KL-divergence regularization function goes to $\infty$ with increasing $w$, it produces MAP estimates $\hat{w}$ that change smoothly with $x$ and $B$ (see Appendix A for more details).

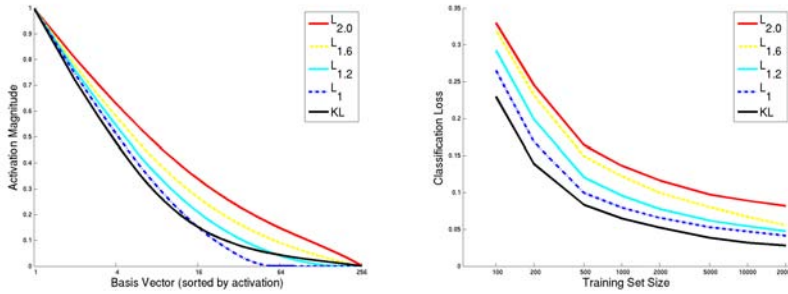

Figure 1: Left: Mean coefficient distribution over the 10,000 digit MNIST test set for various regularization functions. Each example $\hat{w}$ was sorted by magnitude and normalized by $\|\hat{w}\|_\infty$ before computing the mean over all examples. Right: test set classification performance. Regularization functions that produced few large values in each examples (such as KL and L1) performed the best. Forcing small coefficients to be exactly 0 was not necessary for good performance. Note the log scale on the horizontal axis.

| Regularization | Gaussian Noise (Standard Deviation) | | Random Translations (pixels) | |
|---|---|---|---|---|
| | 0.01 | 0.1 | 0.1 | 1 |
| L1 | 0.0283±0.0069 | 0.285±0.056 | 0.138±0.026 | 1.211±0.213 |
| KL | **0.0172±0.0016** | **0.164±0.015** | **0.070±0.011** | **0.671±0.080** |

Table 1: The 10,000 images of handwritten digits in the MNIST test set were used to show the stability benefits of KL-regularization. Distance (in $L_1$) between the representation for $x$, $\hat{w}$, and the representation after adding noise, divided by $\|\hat{w}\|_1$. KL-regularization provides representations that are significantly more stable with respect to both uncorrelated additive Gaussian noise (Left), and correlated noise from translating the digit image in a random direction (Right).

Table 1 quantifies how KL regularization significantly reduces the effect on $\hat{w}$ of adding noise to the input $x$.

This stability improves the usefulness of $\hat{w}$ for prediction. Figure 2 shows the most-discriminative 2-D subspace (as calculated by Multiple Discriminant Analysis [15]) for the input space, the $L_1$ and KL coefficient space, and the KL coefficient space after it has been specialized by back-propagation. The $L_1$ coefficients tame the disorder of the input space so that clusters for each class are apparent, although noisy and overlapping. The switch to KL regularization makes these clusters more distinct, and applying back-propagation further separates the clusters.

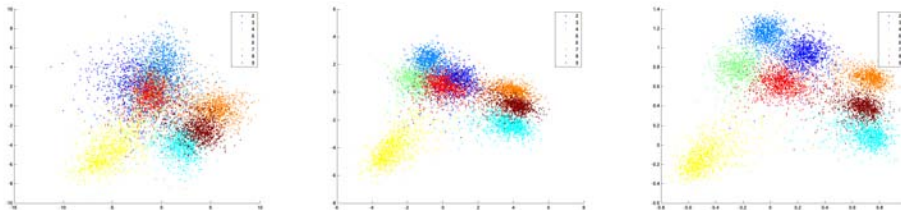

Figure 2: Shown is the distribution of the eight most confusable digit classes in the input space and in the coefficient spaces produced by sparse approximation. Multiple Discriminant Analysis was used to compute the most discriminative 2-D projection of each space. The PCA-whitened input space (left) contains a lot of overlap between the classes. $L_1$ regularization (center) discovers structure in the unlabeled data, but still produces more overlap between classes than KL sparse approximation (right) does with the same basis trained with $L_1$ sparse coding. Figure best seen in color.

## 6.3 Improved Prediction Performance

On all applications, the stability provided by KL-regularization improved performance over $L_1$, and back-propagation further improved performance when the training set had residual error after an output classifier was trained.

### 6.3.1 Handwritten Digit Classification

We tested our algorithm on the benchmark MNIST handwritten digits dataset [16]. 10,000 of the 60,000 training examples were reserved for validation, and classification performance was evaluated on the separate 10,000 example test set. Each example was first reduced to 180D from 768D by PCA, and then sparse coding was performed using a linear transfer function and squared loss[5]. The validation set was used to pick the regularization constant, $\lambda$, and the prior mean for KL, $p$.

Maxent classifiers[6] [17] were then learned on randomly sampled subsets of the training set of various sizes. Switching from $L_1$-regularized to KL-regularized sparse approximation improved performance in all cases (Table 2). When trained on all 50,000 training examples, the test set classification error of KL coefficients, 2.21%, was 37% lower than the 3.53% error rate obtained on the $L_1$-regularized coefficients. As shown in Table 3, this increase in performance was consistent across a diverse set of classification algorithms. After running back-propagation with the KL-prior, the test set error was reduced to 1.30%, which improves on the best results reported[7] for other shallow-architecture permutation-invariant classifiers operating on the same data set without prior knowledge about the problem[8], (see Table 4).

| Training Set Size | 1000 | 2000 | 10000 | 20000 | 50000 |
|---|---|---|---|---|---|
| L1 (Test Set) | 7.72% | 6.63% | 4.74% | 4.16% | 3.53% |
| KL (Test set) | 5.87% | 5.06% | 3.00% | 2.51% | 2.21% |
| KL After Backprop (Test Set) | **5.66** | **4.46**% | **2.31**% | **1.78**% | **1.30**% |
| Improvement from Backprop | 3.6% | 11.9% | 23.0% | 29.1% | 43.0% |
| KL (Training Set) | 0.00% | 0.05% | 1.01% | 1.50% | 1.65% |

Table 2: The ability to optimize the generative model with back-propagation leads to significant performance increases when the training set is not separable by the model learned on the unlabeled data. Shown is the misclassification rate on the MNIST digit classification task. Larger training sets with higher residual error benefit more from back-propagation.

| Classifier | PCA | $L_1$ | KL | KL+backprop |
|---|---|---|---|---|
| Maxent | 7.49% | 3.53% | 2.21% | **1.30%** |
| 2-layer NN | 2.23% | 2.13% | 1.40% | **1.36%** |
| SVM (Linear) | 5.55% | 3.95% | 2.16% | **1.34%** |
| SVM (RBF) | 1.54% | 1.94% | **1.28%** | 1.31% |

Table 3: The stability afforded by the KL-prior improves the performance of all classifier types over the $L_1$ prior. In addition back-propagation allows linear classifiers to do as well as more complicated non-linear classifiers.

| Algorithm | L1 | KL | KL+backprop | SVM | 2-layer NN [18] | 3-layer NN |
|---|---|---|---|---|---|---|
| Test Set Error | 3.53% | 2.21% | **1.30%** | 1.4% | 1.6% | 1.53% |

Table 4: Test set error of various classifiers on the MNIST handwritten digits database.

### 6.3.2 Transfer to Handwritten Character Classification

In [4], a basis learned by $L_1$-regularized sparse coding on handwritten digits was shown to improve classification performance when used for the related problem of handwritten character recognition

with small training data sets ($< 5000$ examples). The handwritten English characters dataset[9] they used consists of 16x8 pixel images of lowercase letters. In keeping with their work, we padded and scaled the images to match the 28x28 pixel size of the MNIST data, projected onto the same PCA basis that was used for the MNIST digits, and learned a basis from the MNIST digits by $L_1$-regularized sparse coding. This basis was then used for sparse approximation of the English characters, along with a linear transfer function and squared loss.

In this application as well, Table 5 shows that simply switching to a KL prior from $L_1$ for sparse approximation significantly improves the performance of a maxent classifier. Furthermore, the KL prior allows online improvement of the sparse coding basis as more labeled data for the character-recognition task becomes available. This improvement increases with the size of the training set, as more information becomes available about the target character recognition task.

| Training Set Size | Raw | PCA | L1 | KL | KL+backprop |
|---|---|---|---|---|---|
| 100 | 44.3 | 46.9 | 44.0 | 49.4 | **50.7** |
| 500 | 60.4 | 61.2 | 63.7 | 69.2 | **69.9** |
| 1000 | 66.3 | 66.7 | 69.5 | 75.0 | **76.4** |
| 5000 | 75.1 | 76.0 | 78.9 | 82.5 | **84.2** |
| 20000 | 79.3 | 79.7 | 83.3 | 86.0 | **89.1** |

Table 5: Classification Accuracy on 26-way English Character classification task.

### 6.3.3 Comparison to sLDA: Movie Review Sentiment Regression

KL-regularized sparse coding bears some similarities to the supervised LDA (sLDA) model introduced in [19], and we provide results for the movie review sentiment classification task [20] used in that work. To match [19] we use vectors of normalized counts for the 5000 words with the highest tf-idf score among the 5006 movie reviews in the data set, use 5-fold cross validation, compute predictions with linear regression on $\hat{w}$, and report our performance in terms of predictive $R^2$ (the fraction of variability in the out-of-fold response values which is captured by the out-of-fold predictions $\hat{y}$: $pR^2 := 1 - (\sum(y - \hat{y})^2)/(\sum(y - \bar{y})^2))$. Since the input is a probability distribution, we use a normalized exponential transfer function, $f(B, w) = \frac{e^{Bw}}{\|e^{Bw}\|_1}$, to compute the reconstruction of the input. For sparse coding we use KL-divergence for both the loss and the regularization functions, as minimizing the KL-divergence between the empirical probability distribution of the document given by each input vector $x$ and $f(B, w)$ is equivalent to maximizing the "constrained Poisson distribution" used to model documents in [21] (details given in appendix D). Table 6 shows that the sparse coding generative model we use is competitive with and perhaps slightly better than LDA. After back-propagation, its performance is superior to the supervised version of LDA, sLDA[10].

| predictive $R^2$ | Algorithm |
|---|---|
| 0.263 | LDA [19] |
| 0.264 | 64D unsupervised KL sparse coding |
| 0.281 | 256D unsupervised KL sparse coding |
| 0.457 | $L_1$-regularized regression [19] |
| 0.500 | sLDA [19] |
| 0.507 | $L_2$-regularized regression |
| **0.534** | 256D KL-regularized coding with backprop |

Table 6: Movie review sentiment prediction task. KL-regularized sparse coding compares favorably with LDA and sLDA.

## 7 Conclusion

This paper demonstrates on a diverse set of applications the advantages of using a differentiable, smooth prior for sparse coding. In particular, a KL-divergence regularization function has significant

advantages over other sparse priors such as $L_1$ because it retains the important aspects of sparsity, while adding stability and differentiability to the MAP estimate $\hat{w}$. Differentiability in particular is shown to lead to state-of-the-art performance by allowing the generative model learned from unlabeled data by sparse-coding to be adapted to a supervised loss function.

### Acknowledgments

David M. Bradley is supported by an NDSEG fellowship provided by the Army Research Office. The authors would also like to thank David Blei, Rajat Raina, and Honglak Lee for their help.

## Footnotes

[1]The maximum likelihood parameter estimate for any regular exponential family distribution can be found by minimizing the corresponding Bregman divergence for that family, and every Bregman divergence has a matching transfer function which leads to a convex minimization problem [8]. That matching transfer function is the gradient $\nabla\phi$ of the function $\phi$ which is associated with the Bregman divergence $D_\phi(x\|y) = \phi(x) - \phi(y) - \langle x - y, \nabla\phi(y)\rangle$.

[2]$L_p^p(x) = \sum_i |x_i|^p$ corresponds to the negative log of a generalized gaussian prior.

[3]$-H(x) = x \log(x)$

[4]In our experiments, if the ratio of backtracking steps to total steps was more than 0.6, $\alpha$ was decreased by 10%. Similarly $\alpha$ was increased by 10% if the ratio fell below 0.3.

[5]This methodology was chosen to match [4]

[6]Also known as multi-class logistic regression

[7]An extensive comparison of classification algorithms for this dataset can be found on the MNIST website, http://yann.lecun.com/exdb/mnist/

[8]Better results have been reported when more prior knowledge about the digit recognition problem is provided to the classifier, either through specialized preprocessing or by giving the classifier a model of how digits are likely to be distorted by expanding the data set with random affine and elastic distortions of the training examples or training with vicinal risk minimization. Convolutional Neural Networks produce the best results on this problem, but they are not invariant to permutations in the input since they contain a strong prior about how pixels are connected.

[9]Available at http://ai.stanford.edu/˜btaskar/ocr/

[10]Given that the word counts used as input are very sparse to begin with, classifiers whose regret bounds depend on the $L_2$ norm of the gradient of the input (such as $L_2$-regularized least squares) do quite well, achieving a predictive $R^2$ value on this application of 0.507.

## References

[1] J. A. Tropp, "Algorithms for simultaneous sparse approximation: part ii: Convex relaxation," *Signal Process.*, vol. 86, no. 3, pp. 589–602, 2006.

[2] B. Olshausen and D. Field, "Sparse coding with an overcomplete basis set: A strategy employed by v1?" *Vision Research*, 1997.

[3] Y. Karklin and M. S. Lewicki, "A hierarchical bayesian model for learning non-linear statistical regularities in non-stationary natural signals," *Neural Computation*, vol. 17, no. 2, pp. 397–423, 2005.

[4] R. Raina, A. Battle, H. Lee, B. Packer, and A. Y. Ng, "Self-taught learning: Transfer learning from unlabeled data," in *ICML '07: Proceedings of the 24th international conference on Machine learning*, 2007.

[5] Y. Bengio, P. Lamblin, D. Popovici, and H. Larochelle, "Greedy layer-wise training of deep networks," in *Advances in Neural Information Processing Systems 19*, B. Schölkopf, J. Platt, and T. Hoffman, Eds. Cambridge, MA: MIT Press, 2007, pp. 153–160.

[6] E. Rietsch, "The maximum entropy approach to inverse problems," *Journal of Geophysics*, vol. 42, pp. 489–506, 1977.

[7] G. Besnerais, J. Bercher, and G. Demoment, "A new look at entropy for solving linear inverse problems," *IEEE Trans. on Information Theory*, vol. 45, no. 5, pp. 1565–1578, July 1999.

[8] A. Banerjee, S. Merugu, I. S. Dhillon, and J. Ghosh, "Clustering with bregman divergences," *Journal of Machine Learning Research*, vol. 6, pp. 1705–1749, 2005.

[9] M. Brand, "Pattern discovery via entropy minimization," in *AISTATS 99*, 1999.

[10] M. Shashanka, B. Raj, and P. Smaragdis, "Sparse overcomplete latent variable decomposition of counts data," in *NIPS*, 2007.

[11] J. Kivinen and M. Warmuth, "Exponentiated gradient versus gradient descent for linear predictors," *Information and Computation*, pp. 1–63, 1997.

[12] N. Cesa-Bianchi and G. Lugosi, *Prediction, Learning, and Games*. Cambridge University Press, 2006.

[13] R. Rifkin and R. Lippert, "Value regularization and fenchel duality," *The Journal of Machine Learning Research*, vol. 8, pp. 441–479, 2007.

[14] D. Widder, *Advanced Calculus*, 2nd ed. Dover Publications, 1989.

[15] R. Duda, P. Hart, and D. Stork, *Pattern classification*. Wiley New York, 2001.

[16] Y. LeCun, L. Bottou, Y. Bengio, and P. Haffner, "Gradient-based learning applied to document recognition," *Proceedings of the IEEE*, vol. 86, no. 11, pp. 2278–2324, 1998.

[17] K. Nigam, J. Lafferty, and A. McCallum, "Using maximum entropy for text classification," 1999. [Online]. Available: citeseer.ist.psu.edu/article/nigam99using.html

[18] P. Y. Simard, D. Steinkraus, and J. C. Platt, "Best practices for convolutional neural networks applied to visual document analysis," in *ICDAR '03: Proceedings of the Seventh International Conference on Document Analysis and Recognition*. Washington, DC, USA: IEEE Computer Society, 2003, p. 958.

[19] D. M. Blei and J. D. McAuliffe, "Supervised topic models," in *NIPS 19*, 2007.

[20] B. Pang and L. Lee, "Seeing stars: Exploiting class relationships for sentiment categorization with respect to rating scales," in *Proceedings of the ACL*, 2005, pp. 115–124.

[21] R. Salakhutdinov and G. Hinton, "Semantic hashing," in *SIGIR workshop on Information Retrieval and applications of Graphical Models*, 2007.

